# The Power of Selective Memory: Self-Bounded Learning of Prediction Suffix Trees

**Ofer Dekel   Shai Shalev-Shwartz   Yoram Singer**
School of Computer Science & Engineering
The Hebrew University, Jerusalem 91904, Israel
{oferd,shais,singer}@cs.huji.ac.il

## Abstract

Prediction suffix trees (PST) provide a popular and effective tool for tasks such as compression, classification, and language modeling. In this paper we take a decision theoretic view of PSTs for the task of sequence prediction. Generalizing the notion of margin to PSTs, we present an online PST learning algorithm and derive a loss bound for it. The depth of the PST generated by this algorithm scales linearly with the length of the input. We then describe a self-bounded enhancement of our learning algorithm which automatically grows a *bounded-depth* PST. We also prove an analogous mistake-bound for the self-bounded algorithm. The result is an efficient algorithm that neither relies on a-priori assumptions on the shape or maximal depth of the target PST nor does it require any parameters. To our knowledge, this is the first provably-correct PST learning algorithm which generates a bounded-depth PST while being competitive with any fixed PST determined in hindsight.

## 1   Introduction

Prediction suffix trees are elegant, effective, and well studied models for tasks such as compression, temporal classification, and probabilistic modeling of sequences (see for instance [13, 11, 7, 10, 2]). Different scientific communities gave different names to variants of prediction suffix trees such as context tree weighting [13] and variable length Markov models [11, 2]. A PST receives an input sequence of symbols, one symbol at a time, and predicts the identity of the next symbol in the sequence based on the most recently observed symbols. Techniques for finding a good prediction tree include online Bayesian mixtures [13], tree growing based on PAC-learning [11], and tree pruning based on structural risk minimization [8]. All of these algorithms either assume an *a-priori* bound on the maximal number of previous symbols which may be used to extend predictions or use a *pre-defined* template-tree beyond which the learned tree cannot grow. Motivated by statistical modeling of biological sequences, Apostolico and Bejerano [1] showed that the bound on the maximal depth can be removed by devising a smart modification of Ron et. al's algorithm [11] (and in fact many other variants), yielding an algorithm with time and space requirements that are linear in the length of the input. However, when modeling very long sequences, both the a-priori bound and the linear space modification might impose serious computational problems.

In this paper we describe a variant of prediction trees for which we are able to devise a learning algorithm that grows bounded-depth trees, while remaining competitive with any fixed prediction tree chosen in hindsight. The resulting time and space requirements of our algorithm are bounded and scale polynomially with the complexity of the best prediction tree. Thus, we are able to sidestep the pitfalls of previous algorithms. The setting we employ is slightly more general than context-based sequence modeling as we assume that we are provided with both an input stream and an output stream. For concreteness, we assume that the input stream is a sequence of vectors $\mathbf{x}_1, \mathbf{x}_2, \ldots$ ($\mathbf{x}_t \in \mathbb{R}^n$) and the output stream is a sequence of symbols $y_1, y_2, \ldots$ over a finite alphabet $\mathcal{Y}$. We denote a sub-sequence $y_i, \ldots, y_j$ of the output stream by $\mathbf{y}_i^j$ and the set of all possible sequences by $\mathcal{Y}^*$. We denote the length of a sequence $\mathbf{s}$ by $|\mathbf{s}|$. Our goal is to correctly predict each symbol in the output stream $y_1, y_2, \ldots$. On each time-step $t$ we predict the symbol $y_t$ based on an arbitrarily long context of previously observed output stream symbols, $\mathbf{y}_1^{t-1}$, and based on the current input vector $\mathbf{x}_t$. For simplicity, we focus on the binary prediction case where $|\mathcal{Y}| = 2$ and for convenience we use $\mathcal{Y} = \{-1, +1\}$ (or $\{-, +\}$ for short) as our output alphabet. Our algorithms and analysis can be adapted to larger output alphabets using ideas from [5].

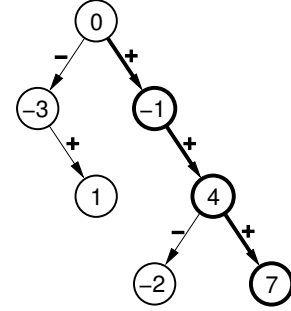

Figure 1: An illustration of the prediction process induced by a PST. The context in this example is: $+++$

The hypotheses we use are confidence-rated and are of the form $h : \mathcal{X} \times \mathcal{Y}^* \to \mathbb{R}$ where the sign of $h$ is the predicted symbol and the magnitude of $h$ is the confidence in this prediction. Each hypothesis is parameterized by a triplet $(\mathbf{w}, \mathcal{T}, g)$ where $\mathbf{w} \in \mathbb{R}^n$, $\mathcal{T}$ is a suffix-closed subset of $\mathcal{Y}^*$ and $g$ is a *context function* from $\mathcal{T}$ into $\mathbb{R}$ ($\mathcal{T}$ is suffix closed if $\forall \mathbf{s} \in \mathcal{T}$ it holds that all of the suffixes of $\mathbf{s}$ are also in $\mathcal{T}$). The prediction extended by a hypothesis $h = (\mathbf{w}, \mathcal{T}, g)$ for the $t$'th symbol is,

$$h(\mathbf{x}_t, \mathbf{y}_1^{t\text{-}1}) = \mathbf{w} \cdot \mathbf{x}_t + \sum_{i:\, \mathbf{y}_{t\text{-}i}^{t\text{-}1} \in \mathcal{T}} 2^{-i/2}\, g\left(\mathbf{y}_{t\text{-}i}^{t\text{-}1}\right) . \tag{1}$$

In words, the prediction is the sum of an inner product between the current input vector $\mathbf{x}_t$ with the weight vector $\mathbf{w}$ and the application of the function $g$ to all the suffixes of the output stream observed thus far that also belong to $\mathcal{T}$. Since $\mathcal{T}$ is a suffix-closed set, it can be described as a rooted tree whose nodes are the sequences constituting $\mathcal{T}$. The children of a node $\mathbf{s} \in \mathcal{T}$ are all the sequences $\sigma\mathbf{s} \in \mathcal{T}$ ($\sigma \in \mathcal{Y}$). Following the terminology of [11], we use the term *prediction suffix tree* (PST) for $\mathcal{T}$ and refer to $\mathbf{s} \in \mathcal{T}$ as a sequence and a node interchangeably. We denote the length of the longest sequence in $\mathcal{T}$ by $\mathrm{depth}(\mathcal{T})$. Given $g$, each node $\mathbf{s} \in \mathcal{T}$ is associated with a value $g(\mathbf{s})$. Note that in the prediction process, the contribution of each context $\mathbf{y}_{t\text{-}i}^{t\text{-}1}$ is multiplied by a factor which is exponentially decreasing in the length of $\mathbf{y}_{t\text{-}i}^{t\text{-}1}$. This type of demotion of long suffixes is common to most PST-based approaches [13, 7, 10] and reflects the a-priori assumption that statistical correlations tend to decrease as the time between events increases. An illustration of a PST where $\mathcal{T} = \{\epsilon, -, +, +-, ++, -++, +++\}$, with the associated prediction for $y_6$ given the context $\mathbf{y}_1^5 = --+++$ is shown in Fig. 1. The predicted value of $y_6$ in the example is $\mathrm{sign}(\mathbf{w} \cdot \mathbf{x}_t + 2^{-1/2} \times (-1) + 2^{-1} \times 4 + 2^{-3/2} \times 7)$. Given $\mathcal{T}$ and $g$ we define the extension of $g$ to all strings over $\mathcal{Y}^*$ by setting $g(\mathbf{s}) = 0$ for $\mathbf{s} \notin \mathcal{T}$. Using this extension, Eq. (1) can be simplified to,

$$h(\mathbf{x}_t, \mathbf{y}_1^{t\text{-}1}) = \mathbf{w} \cdot \mathbf{x}_t + \sum_{i=1}^{t-1} 2^{-i/2}\, g\left(\mathbf{y}_{t\text{-}i}^{t\text{-}1}\right) . \tag{2}$$

We use the online learning loss-bound model to analyze our algorithms. In the online model, learning takes place in rounds. On each round, an instance $\mathbf{x}_t$ is presented to the

online algorithm, which in return predicts the next output symbol. The predicted symbol, denoted $\hat{y}_t$ is defined to be the sign of $h_t(\mathbf{x}_t, \mathbf{y}_1^{t\text{-}1})$. Then, the correct symbol $y_t$ is revealed and with the new input-output pair $(\mathbf{x}_t, y_t)$ on hand, a new hypothesis $h_{t+1}$ is generated which will be used to predict the next output symbol, $y_{t+1}$. In our setting, the hypotheses $h_t$ we generate are of the form given by Eq. (2). Most previous PST learning algorithms employed probabilistic approaches for learning. In contrast, we use a decision theoretic approach by adapting the notion of *margin* to our setting. In the context of PSTs, this approach was first suggested by Eskin in [6]. We define the margin attained by the hypothesis $h_t$ to be $y_t h_t(\mathbf{x}_t, \mathbf{y}_1^{t-1})$. Whenever the current symbol $y_t$ and the output of the hypothesis agree in their sign, the margin is positive. We would like our online algorithm to correctly predict the output stream $y_1, \ldots, y_T$ with a sufficiently large margin of at least 1. This construction is common to many online and batch learning algorithms for classification [12, 4]. Specifically, we use the hinge loss as our margin-based loss function which serves as a proxy for the prediction error. Formally, the hinge loss attained on round $t$ is defined as, $\ell_t = \max \left\{ 0, 1 - y_t h_t \left( \mathbf{x}_t, \mathbf{y}_1^{t\text{-}1} \right) \right\}$. The hinge-loss equals zero when the margin exceeds 1 and otherwise grows linearly as the margin gets smaller. The online algorithms discussed in this paper are designed to suffer small cumulative hinge-loss.

Our algorithms are analyzed by comparing their cumulative hinge-losses and prediction errors with those of any fixed hypothesis $h^\star = (\mathbf{w}^\star, \mathcal{T}^\star, g^\star)$ which can be chosen in hindsight, after observing the entire input and output streams. In deriving our loss and mistake bounds we take into account the complexity of $h^\star$. Informally, the larger $\mathcal{T}^\star$ and the bigger the coefficients of $g^\star(\mathbf{s})$, the more difficult it is to compete with $h^\star$. The squared norm of the context function $g$ is defined as,

$$\|g\|^2 \;\; = \;\; \sum_{\mathbf{s} \in \mathcal{T}} (g(\mathbf{s}))^2 \;\; . \tag{3}$$

The complexity of a hypothesis $h$ (and $h^\star$ in particular) is defined as the sum of $\|\mathbf{w}\|^2$ and $\|g\|^2$. Using the extension of $g$ to $\mathcal{Y}^\star$ we can evaluate $\|g\|^2$ by summing over all $\mathbf{s} \in \mathcal{Y}^\star$.

We present two online algorithms for learning large-margin PSTs. The first incrementally constructs a PST which grows linearly with the length of the input and output sequences, and thus can be arbitrarily large. While this construction is quite standard and similar methods were employed by previous PST-learning algorithms, it provides us with an infrastructure for our second algorithm which grows bounded-depth PSTs. We derive an explicit bound on the maximal depth of the PSTs generated by this algorithm. We prove that both algorithms are competitive with any fixed PST constructed in hindsight. To our knowledge, this is the first provably correct construction of a PST-learning algorithm whose space complexity does not depend on the length of the input-output sequences.

## 2 Learning PSTs of Unbounded Depth

Having described the online prediction paradigm and the form of hypotheses used, we are left with the task of defining the initial hypothesis $h_1$ and the hypothesis update rule. To facilitate our presentation, we assume that all of the instances presented to the online algorithm have a bounded Euclidean norm, namely, $\|\mathbf{x}_t\| \leq 1$. First, we define the initial hypothesis to be $h_1 \equiv 0$. We do so by setting $\mathbf{w}_1 = (0, \ldots, 0)$, $\mathcal{T}_1 = \{\epsilon\}$ and $g_1(\cdot) \equiv 0$. As a consequence, the first prediction always incurs a unit loss. Next, we define the updates applied to the weight vector $\mathbf{w}_t$ and to the PST at the end of round $t$. The weight vector is updated by $\mathbf{w}_{t+1} = \mathbf{w}_t + y_t \tau_t \mathbf{x}_t$, where $\tau_t = \ell_t / (\|\mathbf{x}_t\|^2 + 3)$. Note that if the margin attained on this round is at least 1 then $\ell_t = 0$ and thus $\mathbf{w}_{t+1} = \mathbf{w}_t$. This type of update is common to other online learning algorithms (e.g. [3]). We would like to note in passing that the operation $\mathbf{w}_t \cdot \mathbf{x}_t$ in Eq. (2) can be replaced with an inner product defined via a Mercer kernel. To see this, note that $\mathbf{w}_t$ can be rewritten explicitly as $\sum_{i=1}^{t-1} y_i \tau_i \mathbf{x}_i$ and

**initialize:** $\mathbf{w}_1 = (0, \ldots, 0)$, $\mathcal{T}_1 = \{\epsilon\}$, $g_1(\mathbf{s}) = 0 \;\forall \mathbf{s} \in \mathcal{Y}^{\star}$, $\boxed{P_0 = 0}$

**for** $t = 1, 2, \ldots$ **do**

    Receive an instance $\mathbf{x}_t$ s.t. $\|\mathbf{x}_t\| \leq 1$

    Define: $j = \max\{i \;:\; \mathbf{y}_{t\text{-}i}^{t\text{-}1} \in \mathcal{T}_t\}$

    Calculate: $h_t\left(\mathbf{x}_t, \mathbf{y}_1^{t\text{-}1}\right) = \mathbf{w}_t \cdot \mathbf{x}_t \; + \; \sum_{i=1}^{j} 2^{-i/2} \, g_t\left(\mathbf{y}_{t\text{-}i}^{t\text{-}1}\right)$

    Predict: $\hat{y}_t = \mathrm{sign}\left(h_t\left(\mathbf{x}_t, \mathbf{y}_1^{t\text{-}1}\right)\right)$

    Receive $y_t$ and suffer loss: $\ell_t = \max\left\{0, 1 - y_t h_t\left(\mathbf{x}_t, \mathbf{y}_1^{t\text{-}1}\right)\right\}$

    Set: $\tau_t = \ell_t / \left(\|\mathbf{x}_t\|^2 + 3\right)$ and $d_t = t - 1$

> **if** $(\ell_t \leq 1/2)$ **then**
>     Set: $\tau_t = 0$, $P_t = P_{t-1}$, $d_t = 0$, and continue to the next iteration
> **else**
>     Set: $d_t \;=\; \max\left\{ j, \; \left\lceil 2\log_2\left(2\tau_t\right) - 2\log_2\left(\sqrt{P_{t\text{-}1}^2 + \tau_t \ell_t} - P_{t\text{-}1}\right) \right\rceil \right\}$
>     Set: $P_t = P_{t\text{-}1} + 2\tau_t 2^{-d_t/2}$

    Update weight vector: $\mathbf{w}_{t+1} = \mathbf{w}_t + y_t \tau_t \mathbf{x}_t$

    Update tree:
$$\mathcal{T}_{t+1} \;=\; \mathcal{T}_t \;\cup\; \{\mathbf{y}_{t\text{-}i}^{t\text{-}1} : 1 \leq i \leq d_t\}$$
$$g_{t+1}(\mathbf{s}) \;=\; \begin{cases} g_t(\mathbf{s}) + y_t \, 2^{-|\mathbf{s}|/2} \, \tau_t & \text{if } \mathbf{s} \in \{\mathbf{y}_{t\text{-}i}^{t\text{-}1} : 1 \leq i \leq d_t\} \\ g_t(\mathbf{s}) & \text{otherwise} \end{cases}$$

*(margin: modification required for self-bounded version)*

Figure 2: The online algorithms for learning a PST. The code outside the boxes defines the base algorithm for learning unbounded-depth PSTs. Including the pseudocode inside the boxes gives the self-bounded version.

therefore $\mathbf{w}_t \cdot \mathbf{x}_t = \sum_i y_i \tau_i \, \mathbf{x}_i \cdot \mathbf{x}_t$. Using a kernel operator $K$ simply amounts to replacing the latter expression with $\sum_i y_i \tau_i K(\mathbf{x}_i, \mathbf{x}_t)$.

The update applied to the context function $g_t$ also depends on the scaling factor $\tau_t$. However, $g_t$ is updated only on those strings which participated in the prediction of $\hat{y}_t$, namely strings of the form $\mathbf{y}_{t\text{-}i}^{t\text{-}1}$ for $1 \leq i < t$. Formally, for $1 \leq i < t$ our update takes the form $g_{t+1}(\mathbf{y}_{t\text{-}i}^{t\text{-}1}) = g_t(\mathbf{y}_{t\text{-}i}^{t\text{-}1}) + y_t \, 2^{-i/2} \, \tau_t$. For any other string $\mathbf{s}$, $g_{t+1}(\mathbf{s}) = g_t(\mathbf{s})$. The pseudocode of our algorithm is given in Fig. 2. The following theorem states that the algorithm in Fig. 2 is 2-competitive with any fixed hypothesis $h^{\star}$ for which $\|g^{\star}\|$ is finite.

**Theorem 1.** *Let* $\mathbf{x}_1, \ldots, \mathbf{x}_T$ *be an input stream and let* $y_1, \ldots, y_T$ *be an output stream, where every* $\mathbf{x}_t \in \mathbb{R}^n$, $\|\mathbf{x}_t\| \leq 1$ *and every* $y_t \in \{\text{-}1, 1\}$. *Let* $h^{\star} = (\mathbf{w}^{\star}, \mathcal{T}^{\star}, g^{\star})$ *be an arbitrary hypothesis such that* $\|g^{\star}\| < \infty$ *and which attains the loss values* $\ell_1^{\star}, \ldots, \ell_T^{\star}$ *on the input-output streams. Let* $\ell_1, \ldots, \ell_T$ *be the sequence of loss values attained by the unbounded-depth algorithm in Fig. 2 on the input-output streams. Then it holds that,*

$$\sum_{t=1}^{T} \ell_t^2 \;\leq\; 4\left(\|\mathbf{w}^{\star}\|^2 + \|g^{\star}\|^2\right) \;+\; 2\sum_{t=1}^{T} \left(\ell_t^{\star}\right)^2 \;.$$

*In particular, the above bounds the number of prediction mistakes made by the algorithm.*

*Proof.* For every $t = 1, \ldots, T$ define $\Delta_t = \|\mathbf{w}_t - \mathbf{w}^{\star}\|^2 - \|\mathbf{w}_{t+1} - \mathbf{w}^{\star}\|^2$ and,

$$\hat{\Delta}_t \;=\; \sum_{\mathbf{s} \in \mathcal{Y}^*} \left(g_t(\mathbf{s}) - g^{\star}(\mathbf{s})\right)^2 \;-\; \sum_{\mathbf{s} \in \mathcal{Y}^*} \left(g_{t+1}(\mathbf{s}) - g^{\star}(\mathbf{s})\right)^2 \;. \qquad (4)$$

Note that $\|g_t\|^2$ is finite for any value of $t$ and that $\|g^{\star}\|^2$ is finite due to our assumption, therefore $\hat{\Delta}_t$ is finite and well-defined. We prove the theorem by devising upper and lower

bounds on $\sum_t(\Delta_t + \hat{\Delta}_t)$, beginning with the upper bound. $\sum_t \Delta_t$ is a telescopic sum which collapses to $\|\mathbf{w}_1 - \mathbf{w}^\star\|^2 - \|\mathbf{w}_{t+1} - \mathbf{w}^\star\|^2$. Similarly,

$$\sum_{t=1}^T \hat{\Delta}_t \;=\; \sum_{\mathbf{s}\in\mathcal{Y}^*} \big(g_1(\mathbf{s}) - g^\star(\mathbf{s})\big)^2 \;-\; \sum_{\mathbf{s}\in\mathcal{Y}^*} \big(g_{t+1}(\mathbf{s}) - g^\star(\mathbf{s})\big)^2 \ . \tag{5}$$

Omitting negative terms and using the facts that $\mathbf{w}_1 = (0,\dots,0)$ and $g_1(\cdot) \equiv 0$, we get,

$$\sum_{t=1}^T \left(\Delta_t + \hat{\Delta}_t\right) \;\leq\; \|\mathbf{w}^\star\|^2 \;+\; \sum_{\mathbf{s}\in\mathcal{Y}^*} (g^\star(\mathbf{s}))^2 \;=\; \|\mathbf{w}^\star\|^2 \;+\; \|g^\star\|^2 \ . \tag{6}$$

Having proven an upper bound on $\sum_t(\Delta_t + \hat{\Delta}_t)$, we turn to the lower bound. First, $\Delta_t$ can be rewritten as $\Delta_t = \|\mathbf{w}_t - \mathbf{w}^\star\|^2 - \|(\mathbf{w}_{t+1} - \mathbf{w}_t) + (\mathbf{w}_t - \mathbf{w}^\star)\|^2$ and by expansion of the right-hand term we get that $\Delta_t = -\|\mathbf{w}_{t+1} - \mathbf{w}_t\|^2 - 2(\mathbf{w}_{t+1} - \mathbf{w}_t)\cdot(\mathbf{w}_t - \mathbf{w}^\star)$. Using the value of $\mathbf{w}_{t+1}$ as defined in the update rule of the algorithm ($\mathbf{w}_{t+1} = \mathbf{w}_t + y_t\tau_t\mathbf{x}_t$) gives,

$$\Delta_t \;=\; -\,\tau_t^2\|\mathbf{x}_t\|^2 \;-\; 2\,y_t\,\tau_t\,\mathbf{x}_t\cdot(\mathbf{w}_t - \mathbf{w}^\star) \ . \tag{7}$$

Next, we use similar manipulations to rewrite $\hat{\Delta}_t$. Unifying the two sums that make up $\hat{\Delta}_t$ in Eq. (4) and adding null terms of the form $0 = g_t(\mathbf{s}) - g_t(\mathbf{s})$, we obtain,

$$
\begin{aligned}
\hat{\Delta}_t \;&=\; \textstyle\sum_{\mathbf{s}\in\mathcal{Y}^*}\Big[\big(g_t(\mathbf{s}) - g^\star(\mathbf{s})\big)^2 \;-\; \Big(\big(g_{t+1}(\mathbf{s}) - g_t(\mathbf{s})\big) + \big(g_t(\mathbf{s}) - g^\star(\mathbf{s})\big)\Big)^2\Big] \\
&=\; \textstyle\sum_{\mathbf{s}\in\mathcal{Y}^*}\Big[-\big(g_{t+1}(\mathbf{s}) - g_t(\mathbf{s})\big)^2 \;-\; 2\Big(\big(g_{t+1}(\mathbf{s}) - g_t(\mathbf{s})\big)\big(g_t(\mathbf{s}) - g^\star(\mathbf{s})\big)\Big)\Big] \ .
\end{aligned}
$$

Let $d_t = t - 1$ as defined in Fig. 2. Using the fact that $g_{t+1}$ differs from $g_t$ only on strings of the form $\mathbf{y}_{t\text{-}i}^{t\text{-}1}$, where $g_{t+1}\big(\mathbf{y}_{t\text{-}i}^{t\text{-}1}\big) = g_t\big(\mathbf{y}_{t\text{-}i}^{t\text{-}1}\big) + y_t 2^{-i/2}\tau_t$, we can write $\hat{\Delta}_t$ as,

$$\hat{\Delta}_t \;=\; \sum_{i=1}^{d_t} -2^{\text{-}i}\,\tau_t^2 \;-\; 2\sum_{i=1}^{d_t} y_t\,2^{\text{-}i/2}\,\tau_t\,\Big(g_t\big(\mathbf{y}_{t\text{-}i}^{t\text{-}1}\big) - g^\star\big(\mathbf{y}_{t\text{-}i}^{t\text{-}1}\big)\Big) \ . \tag{8}$$

Summing Eqs. (7-8) gives,

$$
\begin{aligned}
\Delta_t + \hat{\Delta}_t \;&=\; -\tau_t^2\Big(\|\mathbf{x}_t\|^2 + \textstyle\sum_{i=1}^{d_t} 2^{\text{-}i}\Big) \;-\; 2\tau_t\,y_t\Big(\mathbf{w}_t\cdot\mathbf{x}_t + \textstyle\sum_{i=1}^{d_t} 2^{\text{-}i/2}\,g_t\big(\mathbf{y}_{t\text{-}i}^{t\text{-}1}\big)\Big) \\
&\quad + 2\tau_t\,y_t\Big(\mathbf{w}^\star\cdot\mathbf{x}_t + \textstyle\sum_{i=1}^{d_t} 2^{\text{-}i/2}\,g^\star\big(\mathbf{y}_{t\text{-}i}^{t\text{-}1}\big)\Big) \ .
\end{aligned}
\tag{9}
$$

Using $\sum_{i=1}^{d_t} 2^{-i} \leq 1$ with the definitions of $h_t$ and $h^\star$ from Eq. (2), we get that,

$$\Delta_t + \hat{\Delta}_t \;\geq\; -\tau_t^2(\|\mathbf{x}_t\|^2 + 1) \;-\; 2\tau_t\,y_t\,h_t\big(\mathbf{x}_t, \mathbf{y}_1^{t-1}\big) \;+\; 2\tau_t\,y_t\,h^\star\big(\mathbf{x}_t, \mathbf{y}_1^{t-1}\big) \ . \tag{10}$$

Denote the right-hand side of Eq. (10) by $\Gamma_t$ and recall that the loss is defined as $\max\{0, 1 - y_t h_t(\mathbf{x}_t, \mathbf{y}_1^{t\text{-}1})\}$. Therefore, if $\ell_t > 0$ then $-y_t h_t(\mathbf{x}_t, \mathbf{y}_1^{t-1}) = \ell_t - 1$. Multiplying both sides of this equality by $\tau_t$ gives $-\tau_t y_t h_t(\mathbf{x}_t, \mathbf{y}_1^{t-1}) = \tau_t(\ell_t - 1)$. Now note that this equality also holds when $\ell_t = 0$ since then $\tau_t = 0$ and both sides of the equality simply equal zero. Similarly, we have that $y_t h^\star(\mathbf{x}_t, \mathbf{y}_1^{t\text{-}1}) \geq 1 - \ell_t^\star$. Plugging these two inequalities into Eq. (10) gives that,

$$\Gamma_t \;\geq\; -\tau_t^2(\|\mathbf{x}_t\|^2 + 1) + 2\tau_t\,(\ell_t - 1) + 2\tau_t\,(1 - \ell_t^\star) \ ,$$

which in turn equals $-\tau_t^2(\|\mathbf{x}_t\|^2 + 1) + 2\tau_t\,\ell_t - 2\tau_t\,\ell_t^\star$. The lower bound on $\Gamma_t$ still holds if we subtract from it the non-negative term $(2^{1/2}\tau_t - 2^{-1/2}\ell_t^\star)^2$, yielding,

$$
\begin{aligned}
\Gamma_t \;&\geq\; -\tau_t^2(\|\mathbf{x}_t\|^2 + 1) + 2\tau_t\,\ell_t - 2\tau_t\,\ell_t^\star \;-\; \big(2\tau_t^2 - 2\tau_t\ell_t^\star + (\ell_t^\star)^2/2\big) \\
&=\; -\tau_t^2(\|\mathbf{x}_t\|^2 + 3) \;+\; 2\tau_t\,\ell_t \;-\; (\ell_t^\star)^2/2 \ .
\end{aligned}
$$

Using the definition of $\tau_t$ and using the assumption that $\|\mathbf{x}_t\|^2 \leq 1$, we get,

$$\Gamma_t \;\geq\; -\tau_t \ell_t + 2\tau_t \ell_t - \frac{(\ell_t^\star)^2}{2} \;=\; \frac{\ell_t^2}{\|\mathbf{x}_t\|^2 + 3} - \frac{(\ell_t^\star)^2}{2} \;\geq\; \ell_t^2/4 - (\ell_t^\star)^2/2 \;. \quad (11)$$

Since Eq. (10) implies that $\Delta_t + \hat{\Delta}_t \geq \Gamma_t$, summing $\Delta_t + \hat{\Delta}_t$ over all values of $t$ gives,

$$\sum_{t=1}^{T} \left( \Delta_t + \hat{\Delta}_t \right) \;\geq\; \frac{1}{4} \sum_{t=1}^{T} \ell_t^2 \;-\; \frac{1}{2} \sum_{t=1}^{T} (\ell_t^\star)^2 \;.$$

Combining the bound above with Eq. (6) gives the bound stated by the theorem. Finally, we obtain a mistake bound by noting that whenever a prediction mistake occurs, $\ell_t \geq 1$. $\quad\square$

We would like to note that the algorithm for learning unbounded-depth PSTs constructs a sequence of PSTs, $\mathcal{T}_1, \ldots, \mathcal{T}_T$, such that $\mathrm{depth}(\mathcal{T}_t)$ may equal $t$. Furthermore, the number of new nodes added to the tree on round $t$ is on the order of $t$, resulting in $\mathcal{T}_t$ having $O(t^2)$ nodes. However, PST implementation tricks in [1] can be used to reduce the space complexity of the algorithm from quadratic to linear in $t$.

## 3   Self-Bounded Learning of PSTs

The online learning algorithm presented in the previous section has one major drawback, the PSTs it generates can keep growing with each online round. We now describe a modification to the algorithm which casts a limit on the depth of the PST that is learned. Our technique does not rely on arbitrary assumptions on the structure of the tree (e.g. maximal tree depth) nor does it require any parameters. The algorithm determines the depth to which the PST should be updated automatically, and is therefore named the *self-bounded* algorithm for PST learning. The self-bounded algorithm is obtained from the original unbounded algorithm by adding the lines enclosed in boxes in Fig. 2.

A new variable $d_t$ is calculated on every online iteration. On rounds where an update takes place, the algorithm updates the PST up to depth $d_t$, adding nodes if necessary. Below this depth, no nodes are added and the context function is not modified. The definition of $d_t$ is slightly involved, however it enables us to prove that we remain competitive with any fixed hypothesis (Thm. 2) while maintaining a bounded-depth PST (Thm. 3). A point worth noting is that the criterion for performing updates has also changed. Before, the online hypothesis was modified whenever $\ell_t > 0$. Now, an update occurs only when $\ell_t > 1/2$, tolerating small values of loss. Intuitively, this relaxed margin requirement is what enables us to avoid deepening the tree. The algorithm is allowed to predict with lower confidence and in exchange the PST can be kept small. The trade-off between PST size and confidence of prediction is adjusted automatically, extending ideas from [9]. While the following theorem provides a loss bound, this bound can be immediately used to bound the number of prediction mistakes made by the algorithm.

**Theorem 2.** *Let $\mathbf{x}_1, \ldots, \mathbf{x}_T$ be an input stream and let $y_1, \ldots, y_T$ be an output stream, where every $\mathbf{x}_t \in \mathbb{R}^n$, $\|\mathbf{x}_t\| \leq 1$ and every $y_t \in \{-1, 1\}$. Let $h^\star = (\mathbf{w}^\star, \mathcal{T}^\star, g^\star)$ be an arbitrary hypothesis such that $\|g^\star\| < \infty$ and which attains the loss values $\ell_1^\star, \ldots, \ell_T^\star$ on the input-output streams. Let $\ell_1, \ldots, \ell_T$ be the sequence of loss values attained by the self-bounded algorithm in Fig. 2 on the input-output streams. Then the sum of squared-losses attained on those rounds where $\ell_t > 1/2$ is bounded by,*

$$\sum_{t:\ell_t > \frac{1}{2}} \ell_t^2 \;\leq\; \left( \; (1 + \sqrt{5}) \, \|g^\star\| \;+\; 2\, \|\mathbf{w}^\star\| + \left( 2 \sum_{t=1}^{T} (\ell_t^\star)^2 \right)^{1/2} \; \right)^2 \;.$$

*Proof.* We define $\Delta_t$ and $\hat{\Delta}_t$ as in the proof of Thm. 1. First note that the inequality in Eq. (9) in the proof of Thm. 1 still holds. Using the fact that $\sum_{i=1}^{d_t} 2^{-i} \leq 1$ with the definitions of $h_t$ and $h^\star$ from Eq. (2), Eq. (9) becomes,

$$
\begin{aligned}
\Delta_t + \hat{\Delta}_t \geq\ & -\tau_t^2 (\|\mathbf{x}_t\|^2 + 1) - 2\tau_t\, y_t\, h_t\left(\mathbf{x}_t, \mathbf{y}_1^{t-1}\right) + 2\tau_t\, y_t\, h^\star\left(\mathbf{x}_t, \mathbf{y}_1^{t-1}\right) \\
& - 2\tau_t y_t \sum_{i=d_t+1}^{t-1} 2^{-i/2}\, g^\star\left(\mathbf{y}_{t\text{-}i}^{t\text{-}1}\right) \ .
\end{aligned}
\tag{12}
$$

Using the Cauchy-Schwartz inequality we get that

$$
\left| \sum_{i=d_t+1}^{t-1} 2^{-i/2}\, g^\star\left(\mathbf{y}_{t\text{-}i}^{t\text{-}1}\right) \right| \leq \left( \sum_{i=d_t+1}^{t-1} 2^{-i} \right)^{1/2} \left( \sum_{i=d_t+1}^{t-1} \left(g^\star\left(\mathbf{y}_{t\text{-}i}^{t\text{-}1}\right)\right)^2 \right)^{1/2} \leq 2^{-d_t/2}\, \|g^\star\| \ .
$$

Plugging the above into Eq. (12) and using the definition of $\Gamma_t$ from the proof of Thm. 1 gives $\Delta_t + \hat{\Delta}_t \geq \Gamma_t - 2\tau_t 2^{-d_t/2}\, \|g^\star\|$. Using the upper bound on $\Gamma_t$ from Eq. (11) gives,

$$
\Delta_t + \hat{\Delta}_t \geq \tau_t \ell_t - (\ell_t^\star)^2/2 - 2\,\tau_t 2^{-d_t/2}\, \|g^\star\| \ .
\tag{13}
$$

For every $1 \leq t \leq T$, define $L_t = \sum_{i=1}^t \tau_i \ell_i$ and $P_t = \sum_{i=1}^t \tau_i 2^{1-d_i/2}$, and let $P_0 = L_0 = 0$. Summing Eq. (13) over $t$ and comparing to the upper bound in Eq. (6) we get,

$$
L_T \leq \|g^\star\|^2 + \|\mathbf{w}^\star\|^2 + (1/2) \sum_{t=1}^T (\ell_t^\star)^2 + \|g^\star\|\, P_T \ .
\tag{14}
$$

We now use an inductive argument to prove that $P_t \leq \sqrt{L_t}$ for all $0 \leq t \leq T$. This inequality trivially holds for $t = 0$. Assume that $P_{t-1}^2 \leq L_{t-1}$. Expanding $P_t$ we get that

$$
P_t^2 = \left(P_{t-1} + \tau_t 2^{1-d_t/2}\right)^2 = P_{t-1}^2 + P_{t-1}\, 2^{2-d_t/2}\, \tau_t + 2^{2-d_t}\, \tau_t^2 \ .
\tag{15}
$$

We therefore need to show that the right-hand side of Eq. (15) is at most $L_t$. The definition of $d_t$ implies that $2^{-d_t/2}$ is at most $\left((P_{t-1}^2 + \tau_t\ell_t)^{1/2} - P_{t-1}\right)/(2\tau_t)$. Plugging this fact into the right-hand side of Eq. (15) gives that $P_t^2$ cannot exceed $P_{t-1}^2 + \tau_t\ell_t$. Using the inductive assumption $P_{t-1}^2 \leq L_{t-1}$ we get that $P_t^2 \leq L_{t-1} + \tau_t\ell_t = L_t$ and the inductive argument is proven. In particular, we have shown that $P_T \leq \sqrt{L_T}$. Combining this inequality with Eq. (14) we get that

$$
\left(\sqrt{L_T}\right)^2 - \|g^\star\|\, \sqrt{L_T} - \|g^\star\|^2 - \|\mathbf{w}^\star\|^2 - (1/2) \sum_{t=1}^T (\ell_t^\star)^2 \leq 0 \ .
$$

The above equation is a quadratic inequality in $\sqrt{L_T}$ from which it follows that $\sqrt{L_t}$ can be at most as large as the positive root of this equation, namely,

$$
\sqrt{L_T} \leq \frac{1}{2} \left( \|g^\star\| + \left(5\,\|g^\star\|^2 + 4\,\|\mathbf{w}^\star\|^2 + 2 \sum_{t=1}^T (\ell_t^\star)^2\right)^{1/2} \right) \ .
$$

Using the the fact that $\sqrt{a^2 + b^2} \leq (a + b)\ (a, b \geq 0)$ we get that,

$$
\sqrt{L_T} \leq \frac{1 + \sqrt{5}}{2} \|g^\star\| + \|\mathbf{w}^\star\| + \left(\frac{1}{2} \sum_{t=1}^T (\ell_t^\star)^2\right)^{1/2} \ .
\tag{16}
$$

If $\ell_t \leq 1/2$ then $\tau_t\ell_t = 0$ and otherwise $\tau_t\ell_t \geq \ell_t^2/4$. Therefore, the sum of $\ell_t^2$ over the rounds for which $\ell_t > 1/2$ is less than $4\,L_t$, which yields the bound of the theorem. $\qquad\square$

Note that if there exists a fixed hypothesis with $\|g^\star\| < \infty$ which attains a margin of 1 on the entire input sequence, then the bound of Thm. 2 reduces to a constant. Our next theorem states that the algorithm indeed produces bounded-depth PSTs. Its proof is omitted due to the lack of space.

**Theorem 3.** *Under the conditions of Thm. 2, let $\mathcal{T}_1, \ldots, \mathcal{T}_T$ be the sequence of PSTs generated by the algorithm in Fig. 2. Then, for all $1 \leq t \leq T$,*

$$\text{depth}(\mathcal{T}_t) \;\; \leq \;\; 9 \; + \; 2\log_2\Big(2\,\|g^\star\| + \|\mathbf{w}^\star\| + \Big(\frac{1}{2}\sum_{t=1}^{T}(\ell_t^\star)^2\Big)^{1/2} + 1\Big) \;\;.$$

The bound on tree depth given in Thm. 3 becomes particularly interesting when there exists some fixed hypothesis $h^\star$ for which $\sum_t(\ell_t^\star)^2$ is finite and independent of the total length of the output sequence, denoted by $T$. In this case, Thm. 3 guarantees that the depth of the PST generated by the self-bounded algorithm is smaller than a constant which does not depend on $T$. We also would like to emphasize that our algorithm is competitive even with a PST which is deeper than the PST constructed by the algorithm. This can be accomplished by allowing the algorithm's predictions to attain lower confidence than the predictions made by the fixed PST with which it is competing.

**Acknowledgments**   This work was supported by the Programme of the European Community, under the PASCAL Network of Excellence, IST-2002-506778 and by the Israeli Science Foundation grant number 522-04.

# References

[1] G. Bejerano and A. Apostolico. Optimal amnesic probabilistic automata, or, how to learn and classify proteins in linear time and space. *Journal of Computational Biology*, 7(3/4):381–393, 2000.

[2] P. Buhlmann and A.J. Wyner. Variable length markov chains. *The Annals of Statistics*, 27(2):480–513, 1999.

[3] K. Crammer, O. Dekel, S. Shalev-Shwartz, and Y. Singer. Online passive aggressive algorithms. In *Advances in Neural Information Processing Systems 16*, 2003.

[4] N. Cristianini and J. Shawe-Taylor. *An Introduction to Support Vector Machines*. Cambridge University Press, 2000.

[5] O. Dekel, J. Keshet, and Y. Singer. Large margin hierarchical classification. In *Proceedings of the Twenty-First International Conference on Machine Learning*, 2004.

[6] E. Eskin. *Sparse Sequence Modeling with Applications to Computational Biology and Intrusion Detection*. PhD thesis, Columbia University, 2002.

[7] D.P. Helmbold and R.E. Schapire. Predicting nearly as well as the best pruning of a decision tree. *Machine Learning*, 27(1):51–68, April 1997.

[8] M. Kearns and Y. Mansour. A fast, bottom-up decision tree pruning algorithm with near-optimal generalization. In *Proceedings of the Fourteenth International Conference on Machine Learning*, 1996.

[9] P. Auer, N. Cesa-Bianchi and C. Gentile. Adaptive and self-confident on-line learning algorithms. *Journal of Computer and System Sciences*, 64(1):48–75, 2002.

[10] F.C. Pereira and Y. Singer. An efficient extension to mixture techniques for prediction and decision trees. *Machine Learning*, 36(3):183–199, 1999.

[11] D. Ron, Y. Singer, and N. Tishby. The power of amnesia: learning probabilistic automata with variable memory length. *Machine Learning*, 25(2):117–150, 1996.

[12] V.N. Vapnik. *Statistical Learning Theory*. Wiley, 1998.

[13] F.M.J. Willems, Y.M. Shtarkov, and T.J. Tjalkens. The context tree weighting method: basic properties. *IEEE Transactions on Information Theory*, 41(3):653–664, 1995.
